# Convex Neural Networks

**Yoshua Bengio, Nicolas Le Roux, Pascal Vincent, Olivier Delalleau, Patrice Marcotte**
Dept. IRO, Université de Montréal
P.O. Box 6128, Downtown Branch, Montreal, H3C 3J7, Qc, Canada
{bengioy,lerouxni,vincentp,delallea,marcotte}@iro.umontreal.ca

## Abstract

Convexity has recently received a lot of attention in the machine learning community, and the lack of convexity has been seen as a major disadvantage of many learning algorithms, such as multi-layer artificial neural networks. We show that training multi-layer neural networks in which the number of hidden units is learned can be viewed as a convex optimization problem. This problem involves an infinite number of variables, but can be solved by incrementally inserting a hidden unit at a time, each time finding a linear classifier that minimizes a weighted sum of errors.

## 1 Introduction

The objective of this paper is not to present yet another learning algorithm, but rather to point to a previously unnoticed relation between multi-layer neural networks (NNs),Boosting (Freund and Schapire, 1997) and convex optimization. Its main contributions concern the mathematical analysis of an algorithm that is similar to previously proposed incremental NNs, with $L^1$ regularization on the output weights. This analysis helps to understand the underlying convex optimization problem that one is trying to solve.

This paper was motivated by the unproven conjecture (based on anecdotal experience) that when the number of hidden units is "large", the resulting average error is rather insensitive to the random initialization of the NN parameters. One way to justify this assertion is that to really stay stuck in a local minimum, one must have second derivatives positive simultaneously in all directions. When the number of hidden units is large, it seems implausible for none of them to offer a descent direction. Although this paper does not prove or disprove the above conjecture, in trying to do so we found an interesting **characterization of the optimization problem for NNs as a convex program** if the output loss function is convex in the NN output and if the output layer weights are regularized by a convex penalty. More specifically, if the regularization is the $L^1$ norm of the output layer weights, then we show that a "reasonable" solution exists, involving a finite number of hidden units (no more than the number of examples, and in practice typically much less). We present a theoretical algorithm that is reminiscent of Column Generation (Chvátal, 1983), in which hidden neurons are inserted one at a time. Each insertion requires solving a weighted classification problem, very much like in Boosting (Freund and Schapire, 1997) and in particular Gradient Boosting (Mason et al., 2000; Friedman, 2001).

**Neural Networks, Gradient Boosting, and Column Generation**

Denote $\tilde{x} \in \mathbb{R}^{d+1}$ the extension of vector $x \in \mathbb{R}^d$ with one element with value 1. What we call "Neural Network" (NN) here is a predictor for supervised learning of the form $\hat{y}(x) = \sum_{i=1}^{m} w_i h_i(x)$ where $x$ is an input vector, $h_i(x)$ is obtained from a linear discriminant function $h_i(x) = s(v_i \cdot \tilde{x})$ with e.g. $s(a) = \text{sign}(a)$, or $s(a) = \tanh(a)$ or $s(a) = \frac{1}{1+e^{-a}}$. A learning algorithm must specify how to select $m$, the $w_i$'s and the $v_i$'s.

The classical solution (Rumelhart, Hinton and Williams, 1986) involves (a) selecting a loss function $Q(\hat{y}, y)$ that specifies how to penalize for mismatches between $\hat{y}(x)$ and the observed $y$'s (target output or target class), (b) optionally selecting a regularization penalty that favors "small" parameters, and (c) choosing a method to approximately minimize the sum of the losses on the training data $D = \{(x_1, y_1), \ldots, (x_n, y_n)\}$ plus the regularization penalty. Note that in this formulation, an output non-linearity can still be used, by inserting it in the loss function $Q$. Examples of such loss functions are the quadratic loss $||\hat{y} - y||^2$, the hinge loss $\max(0, 1 - y\hat{y})$ (used in SVMs), the cross-entropy loss $-y \log \hat{y} - (1 - y) \log(1 - \hat{y})$ (used in logistic regression), and the exponential loss $e^{-y\hat{y}}$ (used in Boosting).

Gradient Boosting has been introduced in (Friedman, 2001) and (Mason et al., 2000) as a non-parametric greedy-stagewise supervised learning algorithm in which one adds a function at a time to the current solution $\hat{y}(x)$, in a steepest-descent fashion, to form an additive model as above but with the functions $h_i$ typically taken in other kinds of sets of functions, such as those obtained with decision trees. In a stagewise approach, when the $(m+1)$-th basis $h_{m+1}$ is added, only $w_{m+1}$ is optimized (by a line search), like in *matching pursuit* algorithms. Such a greedy-stagewise approach is also at the basis of Boosting algorithms (Freund and Schapire, 1997), which is usually applied using decision trees as bases and $Q$ the exponential loss. It may be difficult to minimize exactly for $w_{m+1}$ and $h_{m+1}$ when the previous bases and weights are fixed, so (Friedman, 2001) proposes to "follow the gradient" in function space, i.e., look for a base learner $h_{m+1}$ that is best correlated with the gradient of the average loss on the $\hat{y}(x_i)$ (that would be the residue $\hat{y}(x_i) - y_i$ in the case of the square loss). The algorithm analyzed here also involves maximizing the correlation between $Q'$ (the derivative of $Q$ with respect to its first argument, evaluated on the training predictions) and the next basis $h_{m+1}$. However, we follow a "stepwise", less greedy, approach, in which all the output weights are optimized at each step, in order to obtain convergence guarantees.

Our approach adapts the Column Generation principle (Chvátal, 1983), a decomposition technique initially proposed for solving linear programs with many variables and few constraints. In this framework, active variables, or "columns", are only generated as they are required to decrease the objective. In several implementations, the column-generation subproblem is frequently a combinatorial problem for which efficient algorithms are available. In our case, the subproblem corresponds to determining an "optimal" linear classifier.

# 2 Core Ideas

Informally, consider the set $\mathcal{H}$ of all possible hidden unit functions (i.e., of all possible hidden unit weight vectors $v_i$). Imagine a NN that has all the elements in this set as hidden units. We might want to impose precision limitations on those weights to obtain either a countable or even a finite set. For such a NN, we only need to learn the output weights. If we end up with a finite number of non-zero output weights, we will have at the end an ordinary feedforward NN. This can be achieved by using a regularization penalty on the output weights that yields sparse solutions, such as the $L^1$ penalty. If in addition the loss function is convex in the output layer weights (which is the case of squared error, hinge loss, $\epsilon$-tube regression loss, and logistic or softmax cross-entropy), then it is easy to show that the overall training criterion is convex in the parameters (which are now only the output weights). The only problem is that there are as many variables in this convex program as there are elements in the set $\mathcal{H}$, which may be very large (possibly infinite). However, we find that with $L^1$ regularization, a finite solution is obtained, and that such a solution can be obtained by greedily inserting one hidden unit at a time. Furthermore, it is theoretically possible to check that the global optimum has been reached.

**Definition 2.1.** *Let $\mathcal{H}$ be a set of functions from an input space $\mathcal{X}$ to $\mathbb{R}$. Elements of $\mathcal{H}$ can be understood as "hidden units" in a NN. Let $\mathcal{W}$ be the Hilbert space of functions from $\mathcal{H}$ to $\mathbb{R}$, with an inner product denoted by $a \cdot b$ for $a, b \in \mathcal{W}$. An element of $\mathcal{W}$ can be understood as the output weights vector in a neural network. Let $h(x) : \mathcal{H} \to \mathbb{R}$ the function that maps any element $h_i$ of $\mathcal{H}$ to $h_i(x)$. $h(x)$ can be understood as the vector of activations*

*of hidden units when input x is observed. Let $w \in \mathcal{W}$ represent a parameter (the output weights). The NN prediction is denoted $\hat{y}(x) = w \cdot h(x)$. Let $Q : \mathbb{R} \times \mathbb{R} \to \mathbb{R}$ be a cost function convex in its first argument that takes a scalar prediction $\hat{y}(x)$ and a scalar target value $y$ and returns a scalar cost. This is the cost to be minimized on example pair $(x, y)$. Let $D = \{(x_i, y_i) : 1 \le i \le n\}$ a training set. Let $\Omega : \mathcal{W} \to \mathbb{R}$ be a convex regularization functional that penalizes for the choice of more "complex" parameters (e.g., $\Omega(w) = \lambda ||w||_1$ according to a 1-norm in $\mathcal{W}$, if $\mathcal{H}$ is countable). We define the* convex NN *criterion $C(\mathcal{H}, Q, \Omega, D, w)$ with parameter $w$ as follows:*

$$C(\mathcal{H}, Q, \Omega, D, w) = \Omega(w) + \sum_{t=1}^{n} Q(w \cdot h(x_t), y_t). \qquad (1)$$

The following is a trivial lemma, but it is conceptually very important as it is the basis for the rest of the analysis in this paper.

**Lemma 2.2.** *The convex NN cost $C(\mathcal{H}, Q, \Omega, D, w)$ is a convex function of $w$.*

*Proof.* $Q(w \cdot h(x_t), y_t)$ is convex in $w$ and $\Omega$ is convex in $w$, by the above construction. $C$ is additive in $Q(w \cdot h(x_t), y_t)$ and additive in $\Omega$. Hence $C$ is convex in $w$. $\qquad \square$

Note that there are no constraints in this convex optimization program, so that at the global minimum all the partial derivatives of $C$ with respect to elements of $w$ cancel.

Let $|\mathcal{H}|$ be the cardinality of the set $\mathcal{H}$. If it is not finite, it is not obvious that an optimal solution can be achieved in finitely many iterations.

Lemma 2.2 says that training NNs from a very large class (with one or more hidden layer) can be seen as convex optimization problems, usually in a very high dimensional space, **as long as we allow the number of hidden units to be selected by the learning algorithm**. By choosing a regularizer that promotes **sparse** solutions, we obtain a solution that has a **finite** number of "active" hidden units (non-zero entries in the output weights vector $w$). This assertion is proven below, in theorem 3.1, for the case of the hinge loss.

However, even if the solution involves a finite number of active hidden units, the convex optimization problem could still be computationally intractable because of the large number of variables involved. One approach to this problem is to apply the principles already successfully embedded in Gradient Boosting, but more specifically in Column Generation (an optimization technique for very large scale linear programs), i.e., add one hidden unit at a time in an incremental fashion. The **important ingredient here is a way to know that we have reached the global optimum, thus not requiring to actually visit all the possible hidden units.** We show that this can be achieved as long as we can solve the sub-problem of finding a linear classifier that minimizes the weighted sum of classification errors. This can be done exactly only on low dimensional data sets but can be well approached using weighted linear SVMs, weighted logistic regression, or Perceptron-type algorithms.

Another idea (not followed up here) would be to consider first a smaller set $\mathcal{H}_1$, for which the convex problem can be solved in polynomial time, and whose solution can theoretically be selected as initialization for minimizing the criterion $C(\mathcal{H}_2, Q, \Omega, D, w)$, with $\mathcal{H}_1 \subset \mathcal{H}_2$, and where $\mathcal{H}_2$ may have infinite cardinality (countable or not). In this way we could show that we can find a solution whose cost satisfies $C(\mathcal{H}_2, Q, \Omega, D, w) \le C(\mathcal{H}_1, Q, \Omega, D, w)$, i.e., is at least as good as the solution of a more restricted convex optimization problem. The second minimization can be performed with a local descent algorithm, without the necessity to guarantee that the global optimum will be found.

## 3 Finite Number of Hidden Neurons

In this section we consider the special case with $Q(\hat{y}, y) = max(0, 1 - y\hat{y})$ the hinge loss, and $L^1$ regularization, and we show that the global optimum of the convex cost involves at most $n + 1$ hidden neurons, using an approach already exploited in (Rätsch, Demiriz and Bennett, 2002) for $L^1$-loss regression Boosting with $L^1$ regularization of output weights.

The training criterion is $C(w) = K\|w\|_1 + \sum_{t=1}^{n} \max{(0, 1 - y_t w \cdot h(x_t))}$. Let us rewrite this cost function as the constrained optimization problem:

$$\min_{w, \xi} \; L(w, \xi) = K\|w\|_1 + \sum_{t=1}^{n} \xi_t \quad \text{s.t.} \; \begin{cases} y_t \left[w \cdot h(x_t)\right] \geq 1 - \xi_t & (C_1) \\ \text{and} \quad \xi_t \geq 0, t = 1, \ldots, n & (C_2) \end{cases}$$

Using a standard technique, the above program can be recast as a linear program. Defining $\lambda = (\lambda_1, \ldots, \lambda_n)$ the vector of Lagrangian multipliers for the constraints $C_1$, its dual problem $(P)$ takes the form (in the case of a finite number $J$ of base learners):

$$(P) : \qquad \max_{\lambda} \sum_{t=1}^{n} \lambda_t \quad \text{s.t.} \; \begin{cases} \lambda \cdot Z_i - K \leq 0, i \in I \\ \text{and} \quad \lambda_t \leq 1, t = 1, \ldots, n \end{cases}$$

with $(Z_i)_t = y_t h_i(x_t)$. In the case of a finite number $J$ of base learners, $I = \{1, \ldots, J\}$. If the number of hidden units is uncountable, then $I$ is a closed bounded interval of $\mathbb{R}$.

Such an optimization problem satisfies all the conditions needed for using Theorem 4.2 from (Hettich and Kortanek, 1993). Indeed:
- $I$ is compact (as a closed bounded interval of $\mathbb{R}$);
- $F : \lambda \mapsto \sum_{t=1}^{n} \lambda_t$ is a concave function (it is even a linear function);
- $g : (\lambda, i) \mapsto \lambda \cdot Z_i - K$ is convex in $\lambda$ (it is actually linear in $\lambda$);
- $\nu(P) \leq n$ (therefore finite) ($\nu(P)$ is the largest value of $F$ satisfying the constraints);
- for every set of $n + 1$ points $i_0, \ldots, i_n \in I$, there exists $\tilde{\lambda}$ such that $g(\tilde{\lambda}, i_j) < 0$ for $j = 0, \ldots, n$ (one can take $\tilde{\lambda} = 0$ since $K > 0$).

Then, from Theorem 4.2 from (Hettich and Kortanek, 1993), the following theorem holds:

**Theorem 3.1.** *The solution of $(P)$ can be attained with constraints $C_2'$ and only $n + 1$ constraints $C_1'$ (i.e., there exists a subset of $n + 1$ constraints $C_1'$ giving rise to the same maximum as when using the whole set of constraints). Therefore, the primal problem associated is the minimization of the cost function of a NN with $n + 1$ hidden neurons.*

# 4 Incremental Convex NN Algorithm

In this section we present a stepwise algorithm to optimize a NN, and show that there is a criterion that allows to verify whether the global optimum has been reached. This is a specialization of minimizing $C(\mathcal{H}, Q, \Omega, D, w)$, with $\Omega(w) = \lambda\|w\|_1$ and $\mathcal{H} = \{h : h(x) = s(v \cdot \tilde{x})\}$ is the set of soft or hard linear classifiers (depending on choice of $s(\cdot)$).

---

**Algorithm ConvexNN($D$,$Q$,$\lambda$,$s$)**

**Input**: training set $D = \{(x_1, y_1), \ldots, (x_n, y_n)\}$, convex loss function $Q$, and scalar regularization penalty $\lambda$. $s$ is either the *sign* function or the *tanh* function.

**(1)** Set $v_1 = (0, 0, \ldots, 1)$ and select $w_1 = \operatorname{argmin}_{w_1} \sum_t Q(w_1 s(1), y_t) + \lambda |w_1|$.

**(2)** Set $i = 2$.

**(3)** Repeat

**(4)**      Let $q_t = Q'(\sum_{j=1}^{i-1} w_j h_j(x_t), y_t)$

**(5)**      If $s = \text{sign}$

**(5a)**        train linear classifier $h_i(x) = \text{sign}(v_i \cdot \tilde{x})$ with examples $\{(x_t, \text{sign}(q_t))\}$ and errors weighted by $|q_t|$, $t = 1 \ldots n$ (i.e., *maximize* $\sum_t q_t h_i(x_t)$)

**(5b)**      else ($s = \tanh$)

**(5c)**        train linear classifier $h_i(x) = \tanh(v_i \cdot \tilde{x})$ to *maximize* $\sum_t q_t h_i(x_t)$.

**(6)**      If $\sum_t q_t h_i(x_t) < \lambda$, **stop**.

**(7)**      Select $w_1, \ldots, w_i$ (and optionally $v_2, \ldots, v_i$) minimizing (exactly or approximately) $C = \sum_t Q(\sum_{j=1}^{i} w_j h_j(x_t), y_t) + \lambda \sum_{j=1}^{i} |w_j|$ such that $\frac{\partial C}{\partial w_j} = 0$ for $j = 1 \ldots i$.

**(8) Return** the predictor $\hat{y}(x) = \sum_{j=1}^{i} w_j h_j(x)$.

---

A key property of the above algorithm is that, at termination, the global optimum is reached, i.e., no hidden unit (linear classifier) can improve the objective. In the case where $s = \text{sign}$, we obtain a Boosting-like algorithm, i.e., it involves finding a classifier which minimizes the weighted cost $\sum_t q_t \text{sign}(v \cdot \tilde{x}_t)$.

**Theorem 4.1.** *Algorithm* **ConvexNN** *stops when it reaches the global optimum of*
$$C(w) = \sum_t Q(w \cdot h(x_t), y_t) + \lambda ||w||_1.$$

*Proof.* Let $w$ be the output weights vector when the algorithm stops. Because the set of hidden units $\mathcal{H}$ we consider is such that when $h$ is in $\mathcal{H}$, $-h$ is also in $\mathcal{H}$, we can assume all weights to be non-negative. By contradiction, if $w' \neq w$ is the global optimum, with $C(w') < C(w)$, then, since $C$ is convex in the output weights, for any $\epsilon \in (0,1)$, we have $C(\epsilon w' + (1-\epsilon)w) \leq \epsilon C(w') + (1-\epsilon)C(w) < C(w)$. Let $w_\epsilon = \epsilon w' + (1-\epsilon)w$. For $\epsilon$ small enough, we can assume all weights in $w$ that are strictly positive to be also strictly positive in $w_\epsilon$. Let us denote by $I_p$ the set of strictly positive weights in $w$ (and $w_\epsilon$), by $I_z$ the set of weights set to zero in $w$ but to a non-zero value in $w_\epsilon$, and by $\delta_{\epsilon k}$ the difference $w_{\epsilon,k} - w_k$ in the weight of hidden unit $h_k$ between $w$ and $w_\epsilon$. We can assume $\delta_{\epsilon j} < 0$ for $j \in I_z$, because instead of setting a small positive weight to $h_j$, one can decrease the weight of $-h_j$ by the same amount, which will give either the same cost, or possibly a lower one when the weight of $-h_j$ is positive. With $o(\epsilon)$ denoting a quantity such that $\epsilon^{-1}o(\epsilon) \rightarrow 0$ when $\epsilon \rightarrow 0$, the difference $\Delta_\epsilon(w) = C(w_\epsilon) - C(w)$ can now be written:

$$
\begin{aligned}
\Delta_\epsilon(w) &= \lambda\left(||w_\epsilon||_1 - ||w||_1\right) + \sum_t \left(Q(w_\epsilon \cdot h(x_t), y_t) - Q(w \cdot h(x_t), y_t)\right) \\[2mm]
&= \lambda\left(\sum_{i \in I_p} \delta_{\epsilon i} + \sum_{j \in I_z} -\delta_{\epsilon j}\right) + \sum_t \sum_k \left(Q'(w \cdot h(x_t), y_t)\delta_{\epsilon k} h_k(x_t)\right) + o(\epsilon) \\[2mm]
&= \sum_{i \in I_p} \left(\lambda \delta_{\epsilon i} + \sum_t q_t \delta_{\epsilon i} h_i(x_t)\right) + \sum_{j \in I_z} \left(-\lambda \delta_{\epsilon j} + \sum_t q_t \delta_{\epsilon j} h_j(x_t)\right) + o(\epsilon) \\[2mm]
&= \sum_{i \in I_p} \delta_{\epsilon i} \frac{\partial C}{\partial w_i}(w) + \sum_{j \in I_z} \left(-\lambda \delta_{\epsilon j} + \sum_t q_t \delta_{\epsilon j} h_j(x_t)\right) + o(\epsilon) \\[2mm]
&= 0 + \sum_{j \in I_z} \left(-\lambda \delta_{\epsilon j} + \sum_t q_t \delta_{\epsilon j} h_j(x_t)\right) + o(\epsilon)
\end{aligned}
$$

since for $i \in I_p$, thanks to step (7) of the algorithm, we have $\frac{\partial C}{\partial w_i}(w) = 0$. Thus the inequality $\epsilon^{-1}\Delta_\epsilon(w) < 0$ rewrites into

$$\sum_{j \in I_z} \epsilon^{-1}\delta_{\epsilon j}\left(-\lambda + \sum_t q_t h_j(x_t)\right) + \epsilon^{-1}o(\epsilon) < 0$$

which, when $\epsilon \rightarrow 0$, yields (note that $\epsilon^{-1}\delta_{\epsilon j}$ does not depend on $\epsilon$ since $\delta_{\epsilon j}$ is linear in $\epsilon$):

$$\sum_{j \in I_z} \epsilon^{-1}\delta_{\epsilon j}\left(-\lambda + \sum_t q_t h_j(x_t)\right) \leq 0 \tag{2}$$

But, $h_i$ being the optimal classifier chosen in step (5a) or (5c), all hidden units $h_j$ verify $\sum_t q_t h_j(x_t) \leq \sum_t q_t h_i(x_t) < \lambda$ and $\forall j \in I_z$, $\epsilon^{-1}\delta_{\epsilon j}\left(-\lambda + \sum_t q_t h_j(x_t)\right) > 0$ (since $\delta_{\epsilon j} < 0$), contradicting eq. 2. $\square$

(Mason et al., 2000) prove a related global convergence result for the AnyBoost algorithm, a non-parametric Boosting algorithm that is also similar to Gradient Boosting (Friedman, 2001). Again, this requires solving as a sub-problem an exact minimization to find a function $h_i \in \mathcal{H}$ that is maximally correlated with the gradient $Q'$ on the output. We now show a simple procedure to select a hyperplane with the best weighted classification error.

**Exact Minimization**

In step (5a) we are required to find a linear classifier that minimizes the weighted sum of classification errors. Unfortunately, this is an NP-hard problem (w.r.t. $d$, see theorem 4 in (Marcotte and Savard, 1992)). However, an exact solution can be easily found in $O(n^3)$ computations for $d = 2$ inputs.

**Proposition 4.2.** *Finding a linear classifier that minimizes the weighted sum of classification error can be achieved in $O(n^3)$ steps when the input dimension is $d = 2$.*

*Proof.* We want to maximize $\sum_i c_i \text{sign}(u \cdot x_i + b)$ with respect to $u$ and $b$, the $c_i$'s being in $\mathbb{R}$. Consider $u$ **fixed** and sort the $x_i$'s according to their dot product with $u$ and denote $r$ the function which maps $i$ to $r(i)$ such that $x_{r(i)}$ is in $i$-th position in the sort. Depending on the value of $b$, we will have $n+1$ possible sums, respectively $-\sum_{i=1}^{k} c_{r(i)} + \sum_{i=k+1}^{n} c_{r(i)}$, $k = 0, \ldots, n$. It is obvious that those sums only depend on the order of the products $u \cdot x_i$, $i = 1, \ldots, n$. When $u$ varies smoothly on the unit circle, as the dot product is a continuous function of its arguments, the changes in the order of the dot products will occur only when there is a pair $(i, j)$ such that $u \cdot x_i = u \cdot x_j$. Therefore, there are at most as many order changes as there are pairs of different points, i.e., $n(n-1)/2$. In the case of $d = 2$, we can enumerate all the different angles for which there is a change, namely $a_1, \ldots, a_z$ with $z \leq \frac{n(n-1)}{2}$. We then need to test at least one $u = [\cos(\theta), \sin(\theta)]$ for each interval $a_i < \theta < a_{i+1}$, and also one $u$ for $\theta < a_1$, which makes a total of $\frac{n(n-1)}{2}$ possibilities. $\square$

It is possible to generalize this result in higher dimensions, and as shown in (Marcotte and Savard, 1992), one can achieve $O(\log(n)n^d)$ time.

---

**Algorithm 1** Optimal linear classifier search

---

$$\textbf{Maximizing } \sum_{i=1}^{n} c_i \delta(\text{sign}(w \cdot x_i), y_i) \textbf{ in dimension 2}$$
**(1)** for $i = 1, \ldots, n$ for $j = i + 1, \ldots, n$
**(3)** $\quad \Theta_{i,j} = \theta(x_i, x_j) + \frac{\pi}{2}$ where $\theta(x_i, x_j)$ is the angle between $x_i$ and $x_j$
**(6)** sort the $\Theta_{i,j}$ in increasing order
**(7)** $w_0 = (1, 0)$
**(8)** for $k = 1, \ldots, \frac{n(n-1)}{2}$
**(9)** $\quad w_k = (\cos \Theta_{i,j}, \sin \Theta_{i,j})$, $u_k = \frac{w_k + w_{k-1}}{2}$
**(10)** $\quad$ sort the $x_i$ according to the value of $u_k \cdot x_i$
**(11)** $\quad$ compute $S(u_k) = \sum_{i=1}^{n} c_i \delta(u_k \cdot x_i), y_i)$
**(12)** output: $\text{argmax}_{u_k} S$

---

**Approximate Minimization**

For data in higher dimensions, the exact minimization scheme to find the optimal linear classifier is not practical. Therefore it is interesting to consider approximate schemes for obtaining a linear classifier with weighted costs. Popular schemes for doing so are the linear SVM (i.e., linear classifier with hinge loss), the logistic regression classifier, and variants of the Perceptron algorithm. In that case, step (5c) of the algorithm is not an exact minimization, and one cannot guarantee that the global optimum will be reached. However, it might be reasonable to believe that finding a linear classifier by minimizing a weighted hinge loss should yield solutions close to the exact minimization. Unfortunately, this is not generally true, as we have found out on a simple toy data set described below. On the other hand, if in step (7) one performs an optimization not only of the output weights $w_j$ ($j \leq i$) but also of the corresponding weight vectors $v_j$, then the algorithm finds a solution close to the global optimum (we could only verify this on 2-D data sets, where the exact solution can be computed easily). It means that at the end of each stage, one first performs a few training iterations of the whole NN (for the hidden units $j \leq i$) with an ordinary gradient descent mechanism (we used conjugate gradients but stochastic gradient descent would work too), optimizing the $w_j$'s and the $v_j$'s, and then one fixes the $v_j$'s and obtains the optimal $w_j$'s for these $v_j$'s (using a convex optimization procedure). In our experiments we used a quadratic

$Q$, for which the optimization of the output weights can be done with a neural network, using the outputs of the hidden layer as inputs.

Let us consider now a bit more carefully what it means to tune the $v_j$'s in step (7). Indeed, changing the weight vector $v_j$ of a selected hidden neuron to decrease the cost is **equivalent to a change in the output weights** $w$**'s**. More precisely, consider the step in which the value of $v_j$ becomes $v'_j$. This is equivalent to the following operation on the $w$'s, when $w_j$ is the corresponding output weight value: the output weight associated with the value $v_j$ of a hidden neuron is set to 0, and the output weight associated with the value $v'_j$ of a hidden neuron is set to $w_j$. This corresponds to an exchange between two variables in the convex program. We are justified to take any such step as long as it allows us to decrease the cost $C(w)$. The fact that we are simultaneously making such exchanges on all the hidden units when we tune the $v_j$'s allows us to move faster towards the global optimum.

**Extension to multiple outputs**

The multiple outputs case is more involved than the single-output case because it is not enough to check the condition $\sum_t h_t q_t > \lambda$. Consider a new hidden neuron whose output is $h_i$ when the input is $x_i$. Let us also denote $\alpha = [\alpha_1, \ldots, \alpha_{n_o}]'$ the vector of output weights between the new hidden neuron and the $n_o$ output neurons. The gradient with respect to $\alpha_j$ is $g_j = \frac{\partial C}{\partial \alpha_j} = \sum_t h_t q_{tj} - \lambda \mathrm{sign}(\alpha_j)$ with $q_{tj}$ the value of the j-th output neuron with input $x_t$. This means that if, for a given $j$, we have $|\sum_t h_t q_{tj}| < \lambda$, moving $\alpha_j$ away from 0 can only increase the cost. Therefore, the right quantity to consider is $(|\sum_t h_t q_{tj}| - \lambda)_+$.

We must therefore find $\mathrm{argmax}_v \sum_j (|\sum_t h_t q_{tj}| - \lambda)^2_+$. As before, this sub-problem is not convex, but it is not as obvious how to approximate it by a convex problem. The stopping criterion becomes: if there is no $j$ such that $|\sum_t h_t q_{tj}| > \lambda$, then all weights must remain equal to 0 and a global minimum is reached.

**Experimental Results**

We performed experiments on the 2-D double moon toy dataset (as used in (Delalleau, Bengio and Le Roux, 2005)), to be able to compare with the exact version of the algorithm. In these experiments, $Q(w \cdot h(x_t), y_t) = [w \cdot h(x_t) - y_t]^2$. The set-up is the following:
• Select a new linear classifier, either (a) the optimal one or (b) an approximate using logistic regression.
• Optimize the output weights using a convex optimizer.
• In case (b), tune both input and output weights by conjugate gradient descent on $C$ and finally re-optimize the output weights using LASSO regression.
• Optionally, remove neurons whose output weight has been set to 0.

Using the approximate algorithm yielded for 100 training examples an average penalized ($\lambda = 1$) squared error of 17.11 (over 10 runs), an average test classification error of 3.68% and an average number of neurons of 5.5 . The exact algorithm yielded a penalized squared error of 8.09, an average test classification error of 5.3%, and required 3 hidden neurons. A penalty of $\lambda = 1$ was nearly optimal for the exact algorithm whereas a smaller penalty further improved the test classification error of the approximate algorithm. Besides, when running the approximate algorithm for a long time, it converges to a solution whose quadratic error is extremely close to the one of the exact algorithm.

# 5   Conclusion

We have shown that training a NN can be seen as a convex optimization problem, and have analyzed an algorithm that can exactly or approximately solve this problem. We have shown that the solution with the hinge loss involved a number of non-zero weights bounded by the number of examples, and much smaller in practice. We have shown that there exists a stopping criterion to verify if the global optimum has been reached, but it involves solving a sub-learning problem involving a linear classifier with weighted errors, which can be com-

putationally hard if the exact solution is sought, but can be easily implemented for toy data sets (in low dimension), for comparing exact and approximate solutions.

The above experimental results are in agreement with our initial conjecture: when there are many hidden units we are much less likely to stall in the optimization procedure, because there are many more ways to descend on the convex cost $C(w)$. They also suggest, based on experiments in which we can compare with the exact sub-problem minimization, that applying Algorithm **ConvexNN** with an approximate minimization for adding each hidden unit **while continuing to tune the previous hidden units** tends to lead to fast convergence to the global minimum. What can get us stuck in a "local minimum" (in the traditional sense, i.e., of optimizing $w$'s and $v$'s together) is simply the **inability to find a new hidden unit weight vector that can improve the total cost (fit and regularization term) even if there exists one**.

Note that as a side-effect of the results presented here, we have a simple way to train **neural networks with hard-threshold hidden units**, since increasing $\sum_t Q'(\hat{y}(x_t), y_t)\text{sign}(v_i x_t)$ can be either achieved exactly (at great price) or approximately (e.g. by using a cross-entropy or hinge loss on the corresponding linear classifier).

### Acknowledgments

The authors thank the following for support: NSERC, MITACS, and the Canada Research Chairs. They are also grateful for the feedback and stimulating exchanges with Sam Roweis, Nathan Srebro, and Aaron Courville.

# References

Chvátal, V. (1983). *Linear Programming*. W.H. Freeman.

Delalleau, O., Bengio, Y., and Le Roux, N. (2005). Efficient non-parametric function induction in semi-supervised learning. In Cowell, R. and Ghahramani, Z., editors, *Proceedings of AISTATS'2005*, pages 96–103.

Freund, Y. and Schapire, R. E. (1997). A decision theoretic generalization of on-line learning and an application to boosting. *Journal of Computer and System Science*, 55(1):119–139.

Friedman, J. (2001). Greedy function approximation: a gradient boosting machine. *Annals of Statistics*, 29:1180.

Hettich, R. and Kortanek, K. (1993). Semi-infinite programming: theory, methods, and applications. *SIAM Review*, 35(3):380–429.

Marcotte, P. and Savard, G. (1992). Novel approaches to the discrimination problem. *Zeitschrift fr Operations Research (Theory)*, 36:517–545.

Mason, L., Baxter, J., Bartlett, P. L., and Frean, M. (2000). Boosting algorithms as gradient descent. In *Advances in Neural Information Processing Systems 12*, pages 512–518.

Rätsch, G., Demiriz, A., and Bennett, K. P. (2002). Sparse regression ensembles in infinite and finite hypothesis spaces. *Machine Learning*.

Rumelhart, D., Hinton, G., and Williams, R. (1986). Learning representations by back-propagating errors. *Nature*, 323:533–536.
